# Effects of Noise on Convergence and Generalization in Recurrent Networks

Kam Jim        Bill G. Horne        C. Lee Giles*

NEC Research Institute, Inc., 4 Independence Way, Princeton, NJ 08540
{kamjim,horne,giles}@research.nj.nec.com

*Also with
UMIACS, University of Maryland, College Park, MD 20742

## Abstract

We introduce and study methods of inserting *synaptic noise* into dynamically-driven recurrent neural networks and show that applying a controlled amount of noise during training may improve convergence and generalization. In addition, we analyze the effects of each noise parameter (additive vs. multiplicative, cumulative vs. non-cumulative, per time step vs. per string) and predict that best overall performance can be achieved by injecting additive noise at each time step. Extensive simulations on learning the dual parity grammar from temporal strings substantiate these predictions.

## 1   INTRODUCTION

There has been much research in applying noise to neural networks to improve network performance. It has been shown that using noisy hidden nodes during training can result in error-correcting codes which increase the tolerance of feedforward nets to unreliable nodes (Judd and Munro, 1992). Also, randomly disabling hidden nodes during the training phase increases the tolerance of MLP's to node failures (Séquin and Clay, 1990). Bishop showed that training with noisy data is equivalent to Tikhonov Regularization and suggested directly minimizing the regularized error function as a practical alternative (Bishop, 1994). Hanson developed a stochastic version of the delta rule which adapt weight means and standard deviations instead

of clean weight values (Hanson, 1990). (Mpitsos and Burton, 1992) demonstrated faster learning rates by adding noise to the weight updates and adapting the magnitude of such noise to the output error. Most relevant to this paper, *synaptic noise* has been applied to MLP's during training to improve fault tolerance and training quality. (Murray and Edwards, 1993)

In this paper, we extend these results by introducing several methods of inserting synaptic noise into recurrent networks, and demonstrate that these methods can improve both convergence and generalization. Previous work on improving these two performance measures have focused on ways of simplifying the network and methods of searching the coarse regions of state space before the fine regions. Our work shows that synaptic noise can improve convergence by searching for *promising regions* of state space, and enhance generalization by enforcing *saturated* states.

## 2   NOISE INJECTION IN RECURRENT NETWORKS

In this paper, we inject noise into a High Order recurrent network (Giles et al., 1992) consisting of $N$ recurrent state neurons $S_j$, $L$ non-recurrent input neurons $I_k$, and $N^2 L$ weights $W_{ijk}$. (For justification of its use see Section 4.) The recurrent network operation is defined by the state process $S_i^{t+1} = g(\sum_{j,k} W_{ijk} S_j^t I_k^t)$, where $g(\cdot)$ is a sigmoid discriminant function. During training, an error function is computed as $E_p = \frac{1}{2} \epsilon_p^2$, where $\epsilon_p = S_O^T - d_p$, $S_O^T$ is the output neuron, and $d_p$ is the target output value for pattern $p$.

Synaptic noise has been simulated on Multi-Layered-Perceptrons by inserting noise to the weights of each layer during training (Murray et al., 1993). Applying this method to recurrent networks is not straightforward because effectively the same weights are propagated forward in time. This can be seen by recalling the BPTT representation of *unrolling* a recurrent network in time into $T$ layers with identical weights, where $T$ is the length of the input string. In Tables 2 and 3, we introduce the noise injection steps for eight recurrent network noise models representing all combinations of the following noise parameters: additive vs. multiplicative, cumulative vs. non-cumulative, per time step vs. per string. As their name imply, additive and multiplicative noise add or multiply the weights by a small noise term. In cumulative noise, the injected noise is accumulated, while in non-cumulative noise the noise from the current step is removed before more noise is injected on the next step. Per time step and per string noise refer to when the noise is inserted: either at each time step or only once for each training string respectively. Table 1 illustrates noise accumulation examples for all additive models (the multiplicative case is analogous).

## 3   ANALYSIS ON THE EFFECTS OF SYNAPTIC NOISE

The effects of each noise model is analyzed by taking the Taylor expansion on the error function around the noise-free weight set. By truncating this expansion to second and lower order terms, we can interpret the effect of noise as a set of regularization terms applied to the error function. From these terms predictions can be made about the effects on generalization and convergence. A similar analysis was

performed on MLP's to demonstrate the effects of synaptic noise on fault tolerance and training quality (Murray et. al., 1993). Tables 2 and 3 list the noise injection step and the resulting first and second order Taylor expansion terms for all noise models. These results are derived by assuming the noise to be zero-mean white with variance $\sigma^2$ and uncorrelated in time.

## 3.1  Predictions on Generalization

One common cause of bad generalization in recurrent networks is the presence of unsaturated state representations. Typically, a network cannot revisit the exact same point in state space, but tends to wander away from its learned state representation. One approach to alleviate this problem is to encourage state nodes to operate in the saturated regions of the sigmoid. The first order error expansion terms of most noise models considered are capable of encouraging the network to achieve saturated states. This can be shown by applying the chain rule to the partial derivative in the first order expansion terms:

$$\frac{\partial S_O^T}{\partial W_{t,ijk}} = \frac{\partial S_O^T}{\partial \Theta_O^T} \sum_l \left[ \left( \frac{\partial \Theta_O^T}{\partial S_l^{T-1}} \frac{\partial S_l^{T-1}}{\partial \Theta_l^{T-1}} \right) \cdots \sum_n \left( \frac{\partial \Theta_m^{t+1}}{\partial S_n^t} \frac{\partial S_n^t}{\partial W_{t,ijk}} \right) \right], \quad (1)$$

where $\Theta_i^t$ is the net input to state node $i$ at time step $t$. The partial derivatives $\frac{\partial S}{\partial \Theta}$ favor internal representations such that the effects of perturbations to the net inputs $\Theta_i^t$ are minimized.

Multiplicative noise implements a form of weight decay because the error expansion terms include the weight products $W_{t,ijk}^2$ or $W_{t,ijk}W_{u,ijk}$. Although weight decay has been shown to improve generalization on feedforward networks (Krogh and Hertz, 1992) we hypothesize this may not be the case for recurrent networks that are learning to solve FSA problems. Large weights are necessary to saturate the state nodes to the upper and lower limits of the sigmoid discriminant function. Therefore, we predict additive noise will allow better generalization because of its absence of weight decay.

Noise models whose first order error term contain the expression $\frac{\partial S_O^T}{\partial W_{t,ijk}} \frac{\partial S_O^T}{\partial W_{u,lmn}}$ will favor saturated states for those partials whose sign correspond to the sign of a majority of the partials. It will favor unsaturated states, operating in the linear region of the sigmoid, for partials whose sign is the minority. Such *sign-dependent enforcement* is not optimal.

The error terms for cumulative per time step noises sum a product with the expression $v\frac{\partial S_O^T}{\partial W_{t,ijk}} \frac{\partial S_O^T}{\partial W_{u,lmn}}$, where $v = min(t+1, u+1)$. The effect of cumulative noise increases more rapidly because of $v$ and thus optimal generalization and detrimental noise effects will occur at lower amplitudes than non-cumulative noise.

For cumulative per string noise models, the products $(t+1)(u+1)$ and $\Psi_{t,ijk}\Psi_{u,lmn}$ in the expansion terms rapidly overwhelm the raw error term. Generalization improvement is not expected for these models.

We also reason that all generalization enhancements will be valid only for a range of noise values, above which noise overwhelms the raw error information.

### 3.2   Predictions on Convergence

Synaptic noise can improve convergence by favoring *promising* weights in the beginning stages of training. This can be demonstrated by examining the second order error expansion term for non-cumulative, multiplicative, per time step noise:

$$\frac{1}{2}\epsilon_p\sigma^2\sum_{t=0}^{T-1}\sum_{ijk}(W_{t,ijk})^2\left(\frac{\partial^2 S_O^T}{\partial(W_{t,ijk})^2}\right).$$

When $\epsilon_p$ is negative, solutions with a negative second order state-weight partial derivative will be de-stabilized. In other words, when the output $S_O^T$ is too small the network will favor updating in a direction such that the first order partial derivative is increasing. A corresponding relationship can be observed for the case when $\epsilon_p$ is positive. Thus the second order term of the error function will allow a higher raw error $\epsilon_p$ to be favored if such an update will place the weights in a more promising area, i.e. a region where weight changes are likely to move $S_O^T$ in a direction to reduce the raw error. The *anticipatory effect* of this term is more important in the beginning stages of training where $\epsilon_p$ is large, and will become insignificant in the finishing stages of training as $\epsilon_p$ approaches zero.

Similar to arguments in Section 3.1, the absence of weight decay will make the learning task easier and improve convergence.

From this discussion it can be inferred that additive per time step noise models should yield the best generalization and convergence performance because of their sign-independent favoring of saturated states and the absence of weight decay. Furthermore, convergence and generalization performance is more *sensitive* to cumulative noise, i.e. optimal performance and detrimental effects will occur at lower amplitudes than in non-cumulative noise.

## 4   SIMULATION RESULTS

In order to perform many experiments in a reasonable amount of computation time, we attempt to learn the simple "hidden-state" dual parity automata from sample strings encoded as temporal sequences. (Dual parity is a 4-state automata that recognizes binary strings containing an even number of ones and zeroes.) We choose a second-order recurrent network since such networks have demonstrated good performance on such problems (Giles et. al., 1992). Thus our experiments consist of 500 simulations for each data point and achieve useful (90%) confidence levels. Experiments are performed with both 3 and 4 state networks, both of which are adequate to learn the automata. The learning rate and momentum are set to 0.5, and the weights are initialized to random values between [-1.0, 1.0]. The data consists of 8191 strings of lengths 0 to 12. The networks are trained on a subset of the training set, called the *working set*, which gradually increases in size until the entire training set is classified correctly. Strings from the working set are presented in *alphabetical order*. The training set consists of the first 1023 strings of lengths 0 to 9, while the initial working set consists of 31 strings of lengths 0 to 4. During testing no noise is added to the weights of the network.

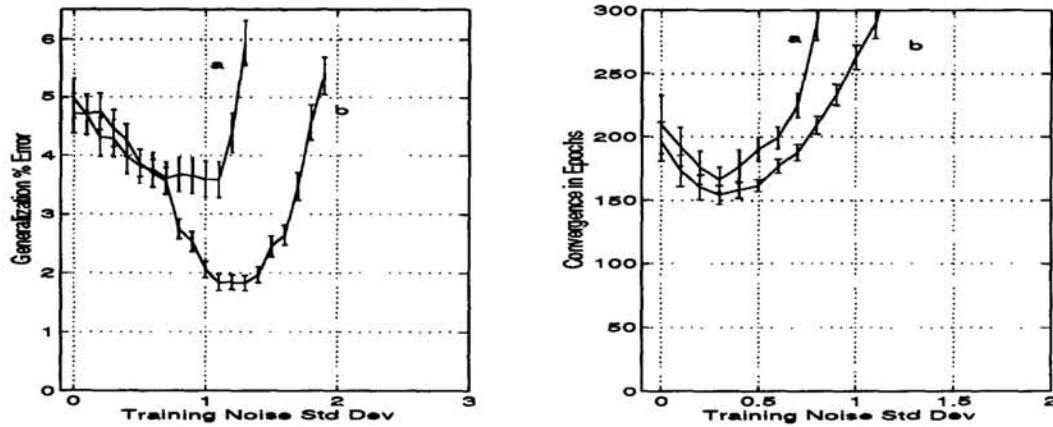

Figure 1: Best Convergence/Generalization for Additive and Multiplicative Noises. (a) multiplicative non-cumulative per time step; (b) additive cumulative per time step.

## 4.1 Convergence and Generalization Performance

Simulated performance closely mirror our predictions. Improvements were observed for all noise models except for cumulative per string noises which failed to converge for all runs. Generalization improvement was more emphasized on networks with 4 states, while convergence enhancement was more noticeable on 3-state networks. The simulations show the following results:

- Additive noise is better tolerated than multiplicative noise, and achieves better convergence and generalization (Figure 1).

- Cumulative noise achieves optimal generalization and convergence at lower amplitudes than non-cumulative noise. Cumulative noise also has a narrower range of beneficial noise, which is defined as the range of noise amplitudes which yields better performance than that of a noiseless network (Figure 2a illustrates this for generalization).

- Per time step noise achieves better convergence/generalization and has a wider range of beneficial values than per string noise (Figure 2b).

Overall, the best performance is obtained by applying cumulative and non-cumulative additive noise at each time step. These results closely match the predictions of section 3.1. The only exceptions are that all multiplicative noise models seem to yield equivalent performance. This discrepancy between prediction and simulation may be due to the detrimental effects of weight decay in multiplicative noise, which can conflict with the advantages of cumulative and per time step noise.

## 4.2 The Payoff Picture: Generalization vs. Convergence

Generalization vs. Convergence results are plotted in Figure 3. Increasing noise amplitudes proceed from the left end-point of each curve to the right end-point.

Table 1: Examples: Additive Noise Accumulation. $\Delta_i$ is the noise at time step $t_i$

| | TIME STEPS | | | |
|---|---|---|---|---|
| NOISE MODEL | $t_1$ | $t_2$ | $t_3$ | $\ldots$ |
| per time step non-cumulative | $W + \Delta_1$ | $W + \Delta_2$ | $W + \Delta_3$ | $\ldots$ |
| per time step cumulative | $W + \Delta_1$ | $W + \Delta_1 + \Delta_2$ | $W + \Delta_1 + \Delta_2 + \Delta_3$ | $\ldots$ |
| per sequence non-cumulative | $W + \Delta_1$ | $W + \Delta_1$ | $W + \Delta_1$ | $\ldots$ |
| per sequence cumulative | $W + \Delta_1$ | $W + 2\Delta_1$ | $W + 3\Delta_1$ | $\ldots$ |

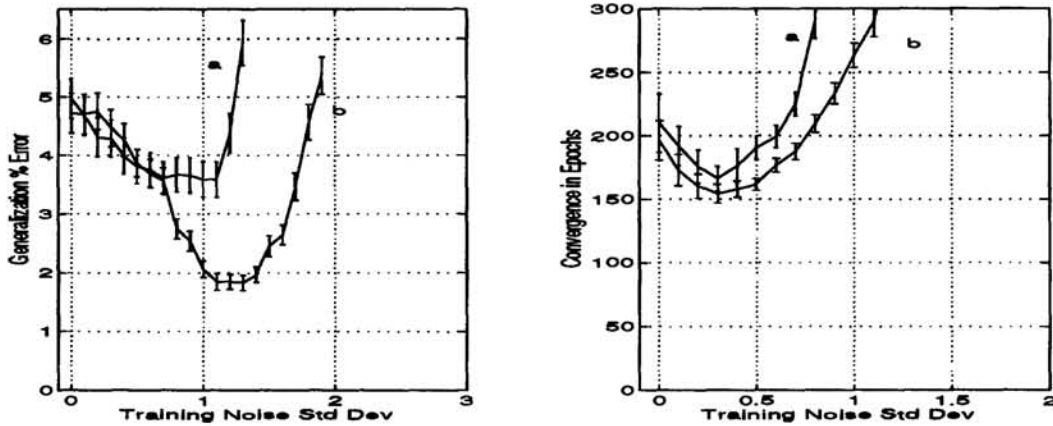

Figure 2: (I) Best Generalization for Cumulative and Non-Cumulative Noises: a) cumulative additive per time step; b) non-cumulative additive per time step. (II) Best Generalization for Per Time Step and Per String Noises: a) non-cumulative per string additive; b) non-cumulative per time step additive.

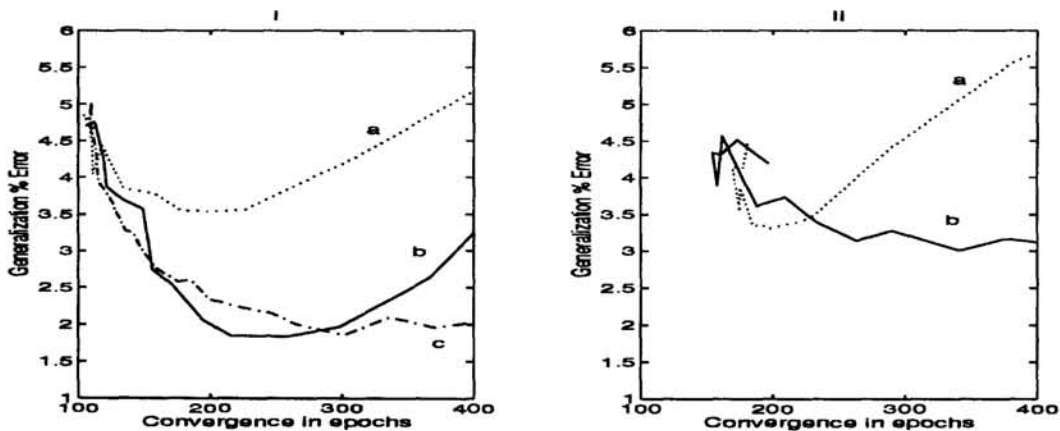

Figure 3: Payoff: Mean Generalization vs. Convergence for 4-state (I) and 3-state(II) recurrent network. (Ia) *Worst 4-state* – non-cumulative multiplicative per string; (Ib, Ic) *Best 4-state* – cumulative and non-cumulative additive per time step, respectively; (IIa) *Worst 3-state* – non-cumulative multiplicative per string ; (IIb) *Best 3-state* – cumulative additive per time step.

Table 2: Noise injection step and error expansion terms for *per time step* noise models. $v = min(t+1, u+1)$. $W^*$ is the noise-free weight set.

| | NON-CUMULATIVE | |
| --- | --- | --- |
| | Additive | Multiplicative |
| Noise step | $W_{t,ijk} = W^*_{ijk} + \Delta_{t,ijk}$ | $W_{t,ijk} = W^*_{ijk} + W^*_{ijk}\Delta_{t,ijk}$ |
| 1st order | $\frac{1}{2}\sigma^2 \sum_{t=0}^{T-1}\sum_{ijk}\left(\frac{\partial S^T_O}{\partial W_{t,ijk}}\right)^2$ | $\frac{1}{2}\sigma^2 \sum_{t=0}^{T-1}\sum_{ijk}\left[W_{t,ijk}\frac{\partial S^T_O}{\partial W_{t,ijk}}\right]^2$ |
| 2nd order | $\frac{1}{2}\epsilon_p\sigma^2 \sum_{t=0}^{T-1}\sum_{ijk}\frac{\partial^2 S^T_O}{\partial (W_{t,ijk})^2}$ | $\frac{1}{2}\epsilon_p\sigma^2 \sum_{t=0}^{T-1}\sum_{ijk}\left[(W_{t,ijk})^2\frac{\partial^2 S^T_O}{\partial (W_{t,ijk})^2}\right]$ |
| | CUMULATIVE | |
| | Additive | Multiplicative |
| Noise step | $W_{t,ijk} = W^*_{ijk} + \sum_{\tau=0}^{t}\Delta_{\tau,ijk}$ | $W_{t,ijk} = W^*_{ijk}\prod_{\tau=0}^{t}\left(1+\Delta_{\tau,ijk}\right)$ |
| 1st order | $\frac{1}{2}\sigma^2 \sum_{t,u=0}^{T-1}\sum_{ijk} v\frac{\partial S^T_O}{\partial W_{t,ijk}}\frac{\partial S^T_O}{\partial W_{u,ijk}}$ | $\frac{1}{2}\sigma^2 \sum_{t,u=0}^{T-1}\sum_{ijk} vW_{t,ijk}W_{u,ijk}\frac{\partial S^T_O}{\partial W_{t,ijk}}\frac{\partial S^T_O}{\partial W_{u,ijk}}$ |
| 2nd order | $\frac{1}{2}\epsilon_p\sigma^2 \sum_{t,u=0}^{T-1}\sum_{ijk} v\frac{\partial^2 S^T_O}{\partial W_{t,ijk}\partial W_{u,ijk}}$ | $\frac{1}{2}\epsilon_p\sigma^2 \sum_{t,u=0}^{T-1}\sum_{ijk} vW_{t,ijk}W_{u,ijk}\frac{\partial^2 S^T_O}{\partial W_{t,ijk}\partial W_{u,ijk}}$ |

Table 3: Noise injection step and error expansion terms for *per string* case. $\Psi_{t,ijk} = W_{t,ijk}\sum_{\tau=0}^{t}c_{t,\tau}(\Delta_{ijk})^{\tau+1}$, $w = t+1$, $x = u+1$. $W^*$ is the noise-free weight set.

| | NON-CUMULATIVE | |
| --- | --- | --- |
| | Additive | Multiplicative |
| Noise step | $W_{t,ijk} = W^*_{ijk} + \Delta_{ijk}$ | $W_{t,ijk} = W^*_{ijk} + W^*_{ijk}\Delta_{ijk}$ |
| 1st order | $\frac{1}{2}\sigma^2 \sum_{t,u=0}^{T-1}\sum_{ijk}\frac{\partial S^T_O}{\partial W_{t,ijk}}\frac{\partial S^T_O}{\partial W_{u,ijk}}$ | $\frac{1}{2}\sigma^2 \sum_{t,u=0}^{T-1}\sum_{ijk}W_{t,ijk}W_{u,ijk}\frac{\partial S^T_O}{\partial W_{t,ijk}}\frac{\partial S^T_O}{\partial W_{u,ijk}}$ |
| 2nd order | $\frac{1}{2}\epsilon_p\sigma^2 \sum_{t,u=0}^{T-1}\sum_{ijk}\frac{\partial^2 S^T_O}{\partial W_{t,ijk}\partial W_{u,ijk}}$ | $\frac{1}{2}\epsilon_p\sigma^2 \sum_{t,u=0}^{T-1}\sum_{ijk}W_{t,ijk}W_{u,ijk}\frac{\partial^2 S^T_O}{\partial W_{t,ijk}W_{u,ijk}}$ |
| | CUMULATIVE | |
| | Additive | Multiplicative |
| Noise step | $W_{t,ijk} = W^*_{ijk} + (t+1)\Delta_{ijk}$ | $W_{t,ijk} = W^*_{ijk}(1+\Delta_{ijk})^t$ $= W^*_{ijk} + \sum_{\tau=0}^{t}c_{t,\tau}(\Delta_{ijk})^{\tau+1}$ |
| 1st order | $\frac{1}{2}\sigma^2 \sum_{t,u=0}^{T-1}\sum_{ijk}wx\frac{\partial S^T_O}{\partial W_{t,ijk}}\frac{\partial S^T_O}{\partial W_{u,ijk}}$ | $\frac{1}{2}\sum_{t,u=0}^{T-1}\sum_{ijk,lmn}\Psi_{t,ijk}\Psi_{u,lmn}\frac{\partial S^T_O}{\partial W_{t,ijk}}\frac{\partial S^T_O}{\partial W_{u,lmn}}$ $+2\epsilon_p^2\sum_{t=0}^{T-1}\sum_{ijk}\Psi_{t,ijk}\frac{\partial S^T_O}{\partial W_{t,ijk}}$ |
| 2nd order | $\frac{1}{2}\epsilon_p\sigma^2 \sum_{t,u=0}^{T-1}\sum_{ijk}wx\frac{\partial^2 S^T_O}{\partial W_{t,ijk}\partial W_{u,ijk}}$ | $\frac{1}{2}\epsilon_p \sum_{t,u=0}^{T-1}\sum_{ijk,lmn}\Psi_{t,ijk}\Psi_{u,lmn}\frac{\partial^2 S^T_O}{\partial w_{t,ijk}\partial W_{u,lmn}}$ |

These plots illustrate the cases where both convergence and generalization are improved. In figure 3II the curves clearly curl down and to the left for lower noise amplitudes before rising to the right at higher noise amplitudes. These lower regions are important because they represent noise values where generalization and convergence improve simultaneously and do not trade off.

## 5   CONCLUSIONS

We have presented several methods of injecting synaptic noise to recurrent neural networks. We summarized the results of an analysis of these methods and empirically tested them on learning the dual parity automaton from strings encoded as temporal sequences. (For a complete discussion of results, see (Jim, Giles, and Horne, 1994) ). Results show that most of these methods can improve generalization and convergence *simultaneously* – most other methods previously discussed in literature cast convergence as a cost for improved generalization performance.

## References

[1] Chris M. Bishop. Training with noise is equivalent to Tikhonov Regularization. *Neural Computation*, 1994. To appear.

[2] Robert M. Burton, Jr. and George J. Mpitsos. Event-dependent control of noise enhances learning in neural networks. *Neural Networks*, 5:627–637, 1992.

[3] C.L. Giles, C.B. Miller, D. Chen, H.H. Chen, G.Z. Sun, and Y.C. Lee. Learning and extracting finite state automata with second-order recurrent neural networks. *Neural Computation*, 4(3):393–405, 1992.

[4] Stephen José Hanson. A stochastic version of the delta rule. *Physica D.*, 42:265–272, 1990.

[5] Kam Jim, C.L. Giles, and B.G. Horne. Synaptic noise in dynamically-driven recurrent neural networks: Convergence and generalization. Technical Report UMIACS-TR-94-89 and CS-TR-3322, Institute for Advanced Computer Studies, University of Maryland, College Park, MD, 1994.

[6] Stephen Judd and Paul W. Munro. Nets with unreliable hidden nodes learn error-correcting codes. In S.J Hanson, J.D. Cowan, and C.L. Giles, editors, *Advances in Neural Information Processing Systems 5*, pages 89–96, San Mateo, CA, 1993. Morgan Kaufmann Publishers.

[7] Anders Krogh and John A. Hertz. A simple weight decay can improve generalization. In J.E. Moody, S.J. Hanson, and R.P. Lippmann, editors, *Advances in Neural Information Processing Systems 4*, pages 450–957, San Mateo, CA, 1992. Morgan Kaufmann Publishers.

[8] Alan F. Murray and Peter J. Edwards. Synaptic weight noise during multilayer perceptron training: Fault tolerance and training improvements. *IEEE Trans. on Neural Networks*, 4(4):722–725, 1993.

[9] Carlo H. Séquin and Reed D. Clay. Fault tolerance in artificial neural networks. In *Proc. of IJCNN*, volume I, pages I–703–708, 1990.